# Robust Nonparametric Regression with Metric-Space valued Output

**Matthias Hein**
Department of Computer Science, Saarland University
Campus E1 1, 66123 Saarbrücken, Germany
hein@cs.uni-sb.de

## Abstract

Motivated by recent developments in manifold-valued regression we propose a family of nonparametric kernel-smoothing estimators with metric-space valued output including several robust versions. Depending on the choice of the output space and the metric the estimator reduces to partially well-known procedures for multi-class classification, multivariate regression in Euclidean space, regression with manifold-valued output and even some cases of structured output learning. In this paper we focus on the case of regression with manifold-valued input and output. We show pointwise and Bayes consistency for all estimators in the family for the case of manifold-valued output and illustrate the robustness properties of the estimators with experiments.

## 1 Introduction

In recent years there has been an increasing interest in learning with output which differs from the case of standard classification and regression. The need for such approaches arises in several applications which possess more structure than the standard scenarios can model. In structured output learning, see [1, 2, 3] and references therein, one generalizes multiclass classification to more general discrete output spaces, in particular incooperating structure of the joint input and output space. These methods have been successfully applied in areas like computational biology, natural language processing and information retrieval. On the other hand there has been a recent series of work which generalizes regression with multivariate output to the case where the output space is a Riemannian manifold, see [4, 5, 6, 7], with applications in signal processing, computer vision, computer graphics and robotics. One can also see this branch as structured output learning if one thinks of a Riemannian manifold as isometrically embedded in a Euclidean space. Then the restriction that the output has to lie on the manifold can be interpreted as constrained regression in Euclidean space, where the constraints couple several output features together.

In this paper we propose a family of kernel estimators for regression with metric-space valued input and output motivated by estimators proposed in [6, 8] for manifold-valued regression. We discuss loss functions and the corresponding Bayesian decision theory for this general regression problem. Moreover, we show that this family of estimators has several well known estimators as special cases for certain choices of the output space and its metric. However, our main emphasis lies on the problem of regression with manifold-valued input and output which includes the multivariate Euclidean case. In particular, we show for all our proposed estimators their pointwise and Bayes consistency, that is in the limit as the sample size goes to infinity the estimated mapping converges to the Bayes optimal mapping. This includes estimators implementing several robust loss functions like the $L_1$-loss, Huber loss or the $\varepsilon$-insensitive loss. This generality is possible since our proof considers directly the functional which is minimized instead of its minimizer as it is usually done in consistency proofs of the Nadaraya-Watson estimator. Finally, we conclude with a toy experiment illustrating the robustness properties and difference of the estimators.

## 2 Bayesian decision theory and loss functions for metric-space valued output

We consider the structured output learning problem where the task is to learn a mapping $\phi : M \to N$ between two metric spaces $M$ and $N$, where $d_M$ denotes the metric of $M$ and $d_N$ the metric of $N$. We assume that both metric spaces $M$ and $N$ are separable[1]. In general, we are in a statistical setting where the given input/output pairs $(X_i, Y_i)$ are i.i.d. samples from a probability measure P on $M \times N$.

In order to prove later on consistency of our metric-space valued estimator we first have to define the Bayes optimal mapping $\phi^* : M \to N$ in the case where $M$ and $N$ are general metric spaces which depends on the employed loss function. In multivariate regression the most common loss function is, $L(y, f(x)) = \|y - f(x)\|_2^2$. However, it is well known that this loss is sensitive to outliers. In univariate regression one therefore uses the $L_1$-loss or other robust loss functions like the Huber or $\varepsilon$-insensitive loss. For the $L_1$-loss the Bayes optimal function $f^*$ is given as $f^*(x) = \text{Med}[Y|X = x]$, where Med denotes the median of $P(Y|X = x)$ which is a robust location measure. Several generalizations of the median for multivariate output have been proposed, see e.g. [9]. In this paper we refer to the minimizer of the loss function $L(y, f(x)) = \|y - f(x)\|_{\mathbb{R}^n}$ resp. $L(y, f(x)) = d_N(y, f(x))$ as the (generalized) median, since this seems to be the only generalization of the univariate median which has a straightforward extension to metric spaces. In analogy to Euclidean case, we will therefore use loss functions penalizing the distance between predicted output and desired output:

$$L(y, \phi(x)) = \Gamma\big(d_N(y, \phi(x))\big), \qquad y \in N, \ x \in M,$$

where $\Gamma : \mathbb{R}_+ \to \mathbb{R}_+$. We will later on restrict $\Gamma$ to a certain family of functions. The associated risk (or expected loss) is: $R_\Gamma(\phi) = \mathbb{E}[L(Y, \phi(X))]$ and its Bayes optimal mapping $\phi_\Gamma^* : M \to N$ can then be determined by

$$
\begin{aligned}
\phi_\Gamma^* &:= \underset{\phi:M \to N, \ \phi \text{ measurable}}{\arg\min} R_\Gamma(\phi) = \underset{\phi:M \to N, \ \phi \text{ measurable}}{\arg\min} \mathbb{E}[\Gamma\big(d_N(Y, \phi(X))\big)] \\
&= \underset{\phi:M \to N, \ \phi \text{ measurable}}{\arg\min} \mathbb{E}_X\big[\mathbb{E}_{Y|X}[\Gamma\big(d_N(Y, \phi(X))\big) \,|\, X]\big].
\end{aligned}
\tag{1}
$$

In the second step we used a result of [10] which states that a joint probability measure on the product of two separable metric spaces can always be factorized into a conditional probability measure and the marginal. In order that the risk is well-defined, we assume that there exists a measurable mapping $\phi : M \to N$ so that $\mathbb{E}[\Gamma\big(d_N(Y, \phi(X))\big)] < \infty$. This holds always once $N$ has bounded diameter. Apart from the global risk $R_\Gamma(\phi)$ we analyze for each $x \in M$ the pointwise risk $R_\Gamma'(x, \phi(x))$,

$$R_\Gamma'(x, \phi(x)) = \mathbb{E}_{Y|X}[\Gamma\big(d_N(Y, \phi(X))\big) \,|\, X = x],$$

which measures the loss suffered by predicting $\phi(x)$ for the input $x \in M$. The total loss $R_\Gamma(\phi)$ of the mapping $\phi$ is then $R_\Gamma(\phi) = \mathbb{E}[R_\Gamma'(X, \phi(X))]$. As in standard regression the factorization allows to find the Bayes optimal mapping $\phi^*$ pointwise,

$$\phi_\Gamma^*(x) = \underset{p \in N}{\arg\min}\, R_\Gamma'(x, p) = \underset{p \in N}{\arg\min}\, \mathbb{E}[\Gamma\big(d_N(Y, p)\big) \,|\, X = x] = \underset{p \in N}{\arg\min} \int_N \Gamma\big(d_N(y, p)\big)\, d\mu_x(y),$$

where $d\mu_x$ is the conditional probability of $Y$ conditioned on $X = x$. Later on we prove consistency for a set of kernel estimators each using a different loss function $\Gamma$ from the following class of functions.

**Definition 1** *A convex function $\Gamma : \mathbb{R}_+ \to \mathbb{R}_+$ is said to be $(\alpha, s)$-bounded if*

- $\Gamma : \mathbb{R}_+ \to \mathbb{R}_+$ *is continuously differentiable, monotonically increasing and $\Gamma(0) = 0$,*

- $\Gamma(2x) \le \alpha\, \Gamma(x)$ *for $x \ge s$ and $\Gamma(s) > 0$ and $\Gamma'(s) > 0$.*

Several functions $\Gamma$ corresponding to standard loss functions in regression are $(\alpha, s)$-bounded:

- $L_p$-type loss: $\Gamma(x) = x^\gamma$ for $\gamma \ge 1$ is $(2^\gamma, 1)$-bounded,

- Huber-loss: $\Gamma(x) = \frac{2x^2}{\varepsilon}$ for $x \le \frac{\varepsilon}{2}$ and $\Gamma(x) = 2x - \frac{\varepsilon}{2}$ for $x > \frac{\varepsilon}{2}$ is $(3, \frac{\varepsilon}{2})$-bounded.

- $\varepsilon$-insensitive loss: $\Gamma(x) = 0$ for $x \leq \varepsilon$ and $\Gamma(x) = x - \varepsilon$ if $x > \varepsilon$ is $(3, 2\varepsilon)$-bounded.

While uniqueness of the minimizer of the pointwise loss functional $R'_\Gamma(x, \cdot)$ cannot be guaranteed anymore in the case of metric space valued output, the following lemma shows that $R'_\Gamma(x, \cdot)$ has reasonable properties (all longer proofs can be found in Section 7 or in the supplementary material). It generalizes a result provided in [11] for $\Gamma(x) = x^2$ to all $(\alpha, s)$-bounded losses.

**Lemma 1** *Let $N$ be a complete and separable metric space such that $d(x, y) < \infty$ for all $x, y \in N$ and every closed and bounded set is compact. If $\Gamma$ is $(\alpha, s)$-bounded and $R'_\Gamma(x, q) < \infty$ for some $q \in N$, then*

- $R'_\Gamma(x, p) < \infty$ *for all $p \in N$,*
- $R'_\Gamma(x, \cdot)$ *is continuous on $N$,*
- *The set of minimizers $Q^* = \underset{q \in N}{\arg\min} \, R'_\Gamma(x, q)$ exists and is compact.*

It is interesting to have a look at one special loss, the case $\Gamma(x) = x^2$. The minimizer of the pointwise risk,

$$F(p) = \underset{p \in N}{\arg\min} \int_N d_N^2(y, p) \, d\mu_x(y),$$

is called the Frechét mean[2] or Karcher mean in the case where $N$ is a manifold. It is the generalization of a mean in Euclidean space to a general metric space. Unfortunately, it needs to be no longer unique as in the Euclidean case. A simple example is the sphere as the output space together with a uniform probability measure on it. In this case every point $p$ on the sphere attains the same value $F(p)$ and thus the global minimum is non-unique. We refer to [12, 13, 11] for more information under which conditions one can prove uniqueness of the global minimizer if $N$ is a Riemannian manifold. The generalization of the median to Riemannian manifolds, that is $\Gamma(x) = x$, is discussed in [9, 4, 8]. For a discussion of the computation of the median in general metric spaces see [14].

## 3   A family of kernel estimators with metric-space valued input and output

In the following we provide the definition of the kernel estimator with metric-space valued output motivated by the two estimators proposed in [6, 8] for manifold-valued output. We use in the following the notation $k_h(x) = \frac{1}{h^m} k(x/h)$.

**Definition 2** *Let $(X_i, Y_i)_{i=1}^l$ be the sample with $X_i \in M$ and $Y_i \in N$. The metric-space-valued kernel estimator $\phi_l : M \to N$ from metric space $M$ to metric space $N$ is defined for all $x \in M$ as*

$$\phi_l(x) = \underset{q \in N}{\arg\min} \frac{1}{l} \sum_{i=1}^l \Gamma\big(d_N(q, Y_i)\big) \, k_h\big(d_M(x, X_i)\big), \qquad (2)$$

*where $\Gamma : \mathbb{R}_+ \to \mathbb{R}_+$ is $(\alpha, s)$-bounded and $k : \mathbb{R}_+ \to \mathbb{R}_+$.*

If the data contains a large fraction of outliers one should use a robust loss function $\Gamma$, see Section 6. Usually the kernel function should be monotonically decreasing since the interpretation of $k_h\big(d_M(x, X_i)\big)$ is to measure the similarity between $x$ and $X_i$ in $M$ which should decrease as the distance increases. The computational complexity to determine $\phi_l(x)$ is quite high as for each test point one has to solve an optimization problem but comparable to structured output learning (see discussion below) where one maximizes for each test point the score function over the output space. For manifold-valued output we will describe in the next section a simple gradient-descent type optimization scheme in order to determine $\phi_l(x)$.

It is interesting to see that several well-known nonparametric estimators for classification and regression can be seen as special cases of this estimator (or a slightly more general form) for different choices of the output space, its metric and the loss function. In particular, the approach shows a certain analogy of a generalization of regression into a continuous space (manifold-valued regression) and regression into a discrete space (structured output learning).

**Multiclass classification:** Let $N = \{1, \ldots, K\}$ where $K$ denotes the number of classes $K$. If there is no special class-structure, then we use the discrete metric on $N$, $d_N(q, q') = 1$ if $q \neq q'$ and 0 else leads for any $\Gamma$ to the standard multiclass classification scheme using a majority vote. Cost-sensitive multiclass classification can be done by using $d_N(q, q')$ to model the cost of misclassifying class $q$ by class $q'$. Since general costs can generally not be modeled by a metric, it should be noted that the estimator can be modified using a similarity function, $s : N \times N \to \mathbb{R}$,

$$\phi_l(x) = \arg\max_{q \in N} \frac{1}{l} \sum_{i=1}^{l} s\big(q, Y_i\big) k_h\big(d_M(x, X_i)\big), \tag{3}$$

The consistency result below can be generalized to this case given that $N$ has finite cardinality.

**Multivariate regression:** Let $N = \mathbb{R}^n$ and $M$ be a metric space. Then for $\Gamma(x) = x^2$, one gets

$$\phi_l(x) = \arg\min_{q \in N} \frac{1}{l} \sum_{i=1}^{l} \|q - Y_i\|^2 \, k_h\big(d_M(x, X_i)\big),$$

which has the solution, $\phi_l(x) = \frac{\frac{1}{l} \sum_{i=1}^{l} k_h\big(d_M(x, X_i)\big) Y_i}{\frac{1}{l} \sum_{i=1}^{l} k_h\big(d_M(x, X_i)\big)}$. This is the well-known Nadaraya-Watson estimator, see [15, 16], on a metric space. In [17] a related estimator is discussed when $M$ is a closed Riemannian manifold and [18] discusses the Nadaraya-Watson estimator when $M$ is a metric space.

**Manifold-valued regression:** In [6] the estimator $\phi_l(x)$ has been proposed for the case where $N$ is a Riemannian manifold and $\Gamma(x) = x^2$, in particular with the emphasis on $N$ being the manifold of shapes. The discussion of a robust median-type estimator, that is $\Gamma(x) = x$, has been done recently in [8]. While it has been shown in [7] that an approach using a global smoothness regularizer outperforms the estimator $\phi_l(x)$, it is a well working baseline with a simple implementation, see Section 4.

**Structured output:** Structured output learning, see [1, 2, 3] and references therein, can be formulated using kernels $k\big((x_1, q_1), (x_2, q_2)\big)$ on the product $M \times N$ of input and output space, which are supposed to measure jointly the similarity and thus can capture non-trivial dependencies between input and output. Using such kernels [1, 2, 3] learn a score function $s : M \times N \to \mathbb{R}$, with

$$\Psi(x) = \arg\max_{q \in N} s(x, q).$$

being the final prediction for $x \in M$. The similarity to our estimator $\phi_l(x)$ in (2) becomes more obvious when we use that in the framework of [1] the learned score function can be written as

$$\Psi_l(x) = \arg\max_{q \in N} \frac{1}{l} \sum_{i=1}^{l} \alpha_i \, k\big((x, q), (X_i, Y_i)\big), \tag{4}$$

where $\alpha \in \mathbb{R}^l$ is the learned coefficient vector. Apart from the coefficient vector $\alpha$ this has almost the form of the previously discussed estimator in Equation (3), using a joint similarity function on input and output space. Clearly, a structured output method where the coefficients $\alpha$ have been optimized, should perform better than $\alpha_i = \text{const}$. In cases where training time is prohibitive the estimator without $\alpha$ is an alternative, at least it provides a useful baseline for structured output learning. Moreover, if the joint kernel factorizes, $k\big((x_1, q_1), (x_2, q_2)\big) = k_M(x_1, x_2) k_N(q_1, q_2)$ on $M$ and $N$, and $k_N(q, q) = \text{const.}$, then one can rewrite the problem in (4) as,

$$\Psi_l(x) = \arg\min_{q \in N} \frac{1}{l} \sum_{i=1}^{l} \alpha_i \, k_M(x, X_i) d_N^2(q, Y_i),$$

where $d_N$ is the induced (semi-)metric[3] of $k_N$. Apart from the learned coefficients this is basically equivalent to $\phi_l(x)$ in (2) for $\Gamma(x) = x^2$.

In the following we restrict ourselves to the case where $M$ and $N$ are Riemannian manifolds. In this case the optimization to obtain $\phi_l(x)$ can still be done very efficiently as the next section shows.

# 4 Implementation of the kernel estimator for manifold-valued output

For fixed $x \in M$, the functional $F(q)$ for $q \in N$ which is optimized in the kernel estimator $\phi_l(x)$ can be rewritten with $w_i = k_h(d_M(x, X_i))$ as,

$$F(q) = \sum_{i=1}^{l} w_i \, \Gamma\big(d_N(q, Y_i)\big).$$

The covariant gradient of $F(q)$ is given as, $\nabla F\big|_q = \sum_{i=1}^{l} w_i \Gamma'\big(d_N(p, Y_i)\big) v_i$, where $v_i \in T_q N$ is a tangent vector at $q$ with $\|v_i\|_{T_q N} = 1$ given by the tangent vector at $q$ of the minimizing[4] geodesic from $Y_i$ to $q$ (pointing "away" from $Y_i$). Denoting by $\exp_q : T_q N \to N$ the exponential map at $q$, the simple gradient descent based optimization scheme can be written as

- choose a random point $q_0$ from $N$,
- while stopping criteria not fulfilled,
    1. compute gradient $\nabla F$ at $q_k$
    2. one has: $q_{k+1} = \exp_{q_k}\big(-\alpha \nabla F|_{q_k}\big)$
    3. determine stepsize $\alpha$ by Armijo rule [19].

As stopping criterion we use either the norm of the gradient or a threshold on the change of $F$. For the experiments in Section 6 we get convergence in 5 to 40 steps.

# 5 Consistency of the kernel estimator for manifold-valued input and output

In this section we show the pointwise and Bayes consistency of the kernel estimator $\phi_l$ in the case where $M$ and $N$ are Riemannian manifolds. This case already subsumes several of the interesting applications discussed in [6, 8]. The proof of consistency of the general metric-space valued kernel estimator (for a restricted class of metric spaces including all Riemannian manifolds) requires high technical overload which is interesting in itself but which would make the paper hard accessible.

The consistency of $\phi_l$ will be proven under the following assumptions:

**Assumptions (A1):**

1. The loss $\Gamma : \mathbb{R}_+ \to \mathbb{R}_+$ is $(\alpha, s)$-bounded.
2. $(X_i, Y_i)_{i=1}^{l}$ is an i.i.d. sample of P on $M \times N$,
3. $M$ and $N$ are compact $m$-and $n$-dimensional manifolds,
4. The data-generating measure P on $M \times N$ is absolutely continuous with respect to the natural volume element,
5. The marginal density on $M$ fulfills: $p(x) \geq p_{\min}, \forall\, x \in M$,
6. The density $p(\cdot, y)$ is continuous on $M$ for all $y \in N$,
7. The kernel fulfills: $a\, 1_{s \leq r_1} \leq k(s) \leq b\, e^{-\gamma\, s^2}$ and $\int_{\mathbb{R}^m} \|x\|\, k(\|x\|)\, dx < \infty$,

Note, that existence of a density is not necessary for consistency. However, in order to keep the proofs simple, we restrict ourselves to this setting. In the following $dV = \sqrt{\det g}\, dx$ denotes the natural volume element of a Riemannian manifold with metric $g$, $\mathrm{vol}(S)$ and $\mathrm{diam}(N)$ are the volume and diameter of the set $S$. For the proof of our main theorem we need the following two propositions. The first one summarizes two results from [20].

**Proposition 1** *Let $M$ be a compact $m$-dimensional Riemannian manifold. Then, there exists $r_0 > 0$ and $S_1, S_2 > 0$ such that for all $x \in M$ the volume of the balls $B(x, r)$ with radius $r \leq r_0$ satisfies,*

$$S_1\, r^m \leq \mathrm{vol}\big(B(x, r)\big) \leq S_2\, r^m.$$

*Moreover, the cardinality $K$ of a $\delta$-covering of $M$ is upper bounded as, $K \leq \frac{\mathrm{vol}(N)}{S_1}\left(\frac{2}{\delta}\right)^m$.*

Moreover, we need a result about convolutions on manifolds.

**Proposition 2** *Let the assumptions A1 hold, then if $f$ is continuous we get for any $x \in M \backslash \partial M$,*

$$\lim_{h \to 0} \int_M k_h(d_M(x,z)) f(z) \, dV(z) = C_x f(x),$$

*where $C_x = \lim_{h \to 0} \int_M k_h(d_M(x,z)) \, dV(z) > 0$. If moreover $f$ is Lipschitz continuous with Lipschitz constant $L$, then there exists a $h_0 > 0$ such that for all $h < h_0(x)$,*

$$\int_M k_h(d_M(x,z)) f(z) \, dV(z) = C_x \, f(x) + O(h).$$

The following main theorem proves the almost sure pointwise convergence of the manifold-valued kernel estimator for all $(\alpha, s)$-bounded loss functions $\Gamma$.

**Theorem 1** *Suppose the assumptions in A1 hold. Let $\phi_l(x)$ be the estimate of the kernel estimator for sample size $l$. If $h \to 0$ and $lh^m / \log l \to \infty$, then for any $x \in M \backslash \partial M$,*

$$\lim_{l \to \infty} |R'_\Gamma(x, \phi_l(x)) - \arg\min_{q \in N} R'_\Gamma(x,q)| = 0, \quad almost \ surely.$$

*If additionally $p(\cdot, y)$ is Lipschitz-continuous for any $y \in N$, then*

$$\lim_{l \to \infty} |R'_\Gamma(x, \phi_l(x)) - \arg\min_{q \in N} R'_\Gamma(x,q)| = O(h) + O\big(\sqrt{\log l / (l \, h^m)}\big), \quad almost \ surely.$$

*The optimal rate is given by $h = O\big((\log l / l)^{\frac{1}{2+m}}\big)$ so that*

$$\lim_{l \to \infty} R'_\Gamma(x, \phi_l(x)) - \arg\min_{q \in N} R'_\Gamma(x,q) = O\Big(\big(\log l / l\big)^{\frac{1}{2+m}}\Big), \quad almost \ surely.$$

Note, that the condition $l \, h^m / \log l \to \infty$ for convergence is the same as for the Nadaraya-Watson estimator on a $m$-dimensional Euclidean space. This had to be expected as this condition still holds if one considers multivariate output, see [15, 16]. Thus, doing regression with manifold-valued output is not more "difficult" than standard regression with multivariate output.

Next, we show Bayes consistency of the manifold-valued kernel estimator.

**Theorem 2** *Let the assumptions A1 hold. If $h \to 0$ and $lh^m / \log l \to \infty$, then*

$$\lim_{l \to \infty} R_\Gamma(\phi_l) - R_\Gamma(\phi^*) = 0, \quad almost \ surely.$$

**Proof:** We have, $R_\Gamma(\phi_l) - R_\Gamma(\phi^*) \leq \mathbb{E}[|R'_\Gamma(X, \phi_l(X)) - R'_\Gamma(X, \phi^*(X))|]$. Moreover, we have almost everywhere, $\lim_{l \to \infty} R'_\Gamma(x, \phi_l(x)) = R'_\Gamma(x, \phi^*(x))$ almost surely. Since $\mathbb{E}[R'_\Gamma(X, \phi(X))] < \infty$ and $\mathbb{E}[R'_\Gamma(X, \phi^*(X))] < \infty$, an extension of the dominated convergence theorem proven by Glick, see [21], provides the result. $\square$

## 6 Experiments

We illustrate the differences of median and mean type estimator on a synthetic dataset with the task of estimating a curve on the sphere, that is $M = [0,1]$ and $N = S^1$. The kernel used had the form, $k\big(|x - y|/h\big) = 1 - |x - y|/h$. The parameter $h$ was found by 5-fold cross validation from the set $[5, 10, 20, 40] * 10^{-3}$. The results are summarized for different levels of outliers and different levels of van-Mises noise (note that the parameter $k$ is inverse to the variance of the distribution) in Table 1. As expected the the $L_1$-loss and the Huber loss as robust loss functions outperform the $L_2$-loss in the presence of outliers, whereas the $L_2$-loss outperforms the robust versions when no outliers are present. Note, that the Huber loss as a hybrid version between $L_1$- and $L_2$-loss is even slightly better than the $L_1$-loss in the presence of outliers as well as in the outlier free case. Thus for a given dataset it makes sense not only to do cross-validation of the parameter $h$ of the kernel function but also over different loss functions in order to adapt to possible outliers in the data.

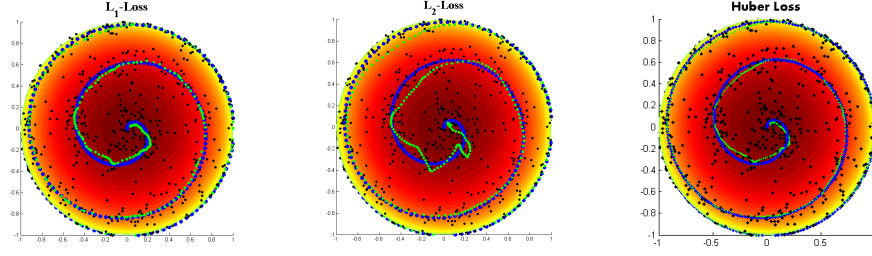

Figure 1: Regression problem on the sphere with 1000 training points (black points). The blue points are the ground truth disturbed by van Mises noise with parameter $k = 100$ and $20\%$ (outliers) with $k = 3$. The estimated curves are shown in green. **Left:** Result of $L_1$-loss, mean error (ME) 0.256, mean squared error (MSE) 0.165. **Middle:** Result of $L_2$-loss: ME $= 0.265$, MSE $= 0.169$. **Right:** Result of Huber loss with $\varepsilon = 0.1$: ME $= 0.255$, MSE $= 0.165$. In particular, the curves found using $L_1$ and Huber loss are very close to the ground truth.

Table 1: Mean squared error (unit $10^{-1}$) for regression on the sphere - for different noise levels $k$, number of labeled points, without and with outliers. Results are averaged over 10 runs.

| | | no outliers | | | 20% outliers | | |
|---|---|---|---|---|---|---|---|
| Number of samples | | 100 | 500 | 1000 | 100 | 500 | 1000 |
| $L_1$-Loss | $k = 100$ | $0.63 \pm 0.11$ | $0.260 \pm 0.027$ | $0.219 \pm 0.003$ | $2.1 \pm 0.2$ | $\mathbf{1.57 \pm 0.05}$ | $1.521 \pm 0.015$ |
| $\Gamma(x) = x$ | $k = 1000$ | $0.43 \pm 0.12$ | $0.043 \pm 0.005$ | $0.030 \pm 0.001$ | $2.1 \pm 0.5$ | $1.45 \pm 0.03$ | $1.400 \pm 0.008$ |
| $L_2$-Loss | $k = 100$ | $\mathbf{0.43 \pm 0.10}$ | $\mathbf{0.230 \pm 0.007}$ | $\mathbf{0.208 \pm 0.001}$ | $\mathbf{2.0 \pm 0.2}$ | $1.59 \pm 0.02$ | $1.549 \pm 0.021$ |
| $\Gamma(x) = x^2$ | $k = 1000$ | $\mathbf{0.28 \pm 0.16}$ | $\mathbf{0.032 \pm 0.003}$ | $\mathbf{0.025 \pm 0.001}$ | $\mathbf{2.0 \pm 0.4}$ | $1.51 \pm 0.03$ | $1.447 \pm 0.015$ |
| Huber-Loss | $k = 100$ | $0.61 \pm 0.11$ | $0.257 \pm 0.026$ | $0.218 \pm 0.003$ | $2.1 \pm 0.2$ | $\mathbf{1.57 \pm 0.05}$ | $\mathbf{1.520 \pm 0.021}$ |
| with $\varepsilon = 0.1$ | $k = 1000$ | $0.42 \pm 0.12$ | $0.040 \pm 0.005$ | $0.028 \pm 0.001$ | $2.1 \pm 0.5$ | $\mathbf{1.44 \pm 0.02}$ | $\mathbf{1.397 \pm 0.008}$ |

## 7 Proofs

**Lemma 2** *Let $\phi : \mathbb{R}_+ \to \mathbb{R}$ be convex, differentiable and monotonically increasing. Then*
$$\min\{\phi'(x), \phi'(y)\}|y - x| \leq |\phi(y) - \phi(x)| \leq \max\{\phi'(x), \phi'(y)\}|y - x|.$$

**Proof of Theorem 1**  We define $R'_{\Gamma,l}(x, q) = \frac{\frac{1}{l}\sum_{i=1}^{l} \Gamma(d_N(q, Y_i)) \, k_h(d_M(x, X_i))}{\mathbb{E}[k_h(d_M(x, X))]}$. Note that $\phi_l(x) = \arg\min_{q \in N} R'_{\Gamma,l}(x, q)$ as we have only divided by a constant factor. We use the standard technique for the pointwise estimate,

$$R'_{\Gamma}(x, \phi_l(x)) - \min_{q \in N} R'_{\Gamma}(x, q) \leq R'_{\Gamma}(x, \phi_l(x)) - R'_{\Gamma,l}(x, \phi_l(x)) + R'_{\Gamma,l}(x, \phi_l(x)) - \min_{q \in N} R'_{\Gamma}(x, q)$$

$$\leq 2 \sup_{q \in N} |R'_{\Gamma,l}(x, q) - R'_{\Gamma}(x, q)|.$$

In order to bound the supremum, we will work on the event $\mathcal{E}$, where we assume, $\left|\frac{\frac{1}{l}\sum_{i=1}^{l} k_h(d_M(x, X_i))}{\mathbb{E}[k_h(d_M(x, X))]} - 1\right| < \frac{1}{2}$, which holds with probability $1 - 2\,e^{-C\,l\,h^m}$ for some constant $C$. Moreover, we assume to have a $\delta$-covering of $N$ with centers $\mathcal{N}_\delta = \{q_\alpha\}_{\alpha=1}^{K}$ where using Lemma 1 we have $K \leq \frac{\text{vol}(N)}{S_1}\left(\frac{2}{\delta}\right)^n$. Thus for each $q \in N$ there exists $q_\alpha \in \mathcal{N}_\delta$ such that $d_N(q, q_\alpha) \leq \delta$. Introducing $R_\Gamma^E(x, q) = \frac{\mathbb{E}[\Gamma(d_N(q, Y)) k_h(d_M(x, X))]}{\mathbb{E}[k_h(d_M(x, X))]}$ and using the decomposition,

$$R'_{\Gamma,l}(x, q) - R'_{\Gamma}(x, q) = R'_{\Gamma,l}(x, q) - R'_{\Gamma,l}(x, q_\alpha) + R'_{\Gamma,l}(x, q_\alpha) - R_\Gamma^E(x, q_\alpha)$$
$$+ R_\Gamma^E(x, q_\alpha) - R_\Gamma^E(x, q) + R_\Gamma^E(x, q) - R'_{\Gamma}(x, q),$$

we have to control four terms,

$$\left|R'_{\Gamma,l}(x, q) - R'_{\Gamma,l}(x, q_\alpha)\right| = \left|\frac{\frac{1}{l}\sum_{i=1}^{l}\left(\Gamma(d_N(q, Y_i)) - \Gamma(d_N(q_\alpha, Y_i))\right) k_h(d_M(x, X_i))}{\mathbb{E}[k_h(d_M(x, X))]}\right|$$

$$\leq 2\, d_N(q, q_\alpha)\, \Gamma'\left(\text{diam}(N)\right) \frac{\frac{1}{l}\sum_{i=1}^{l} k_h(d_M(x, X_i))}{\mathbb{E}[k_h(d_M(x, X))]} \leq 3\,\Gamma'\left(\text{diam}(N)\right)\delta.$$

where we have used Lemma 2 and the fact that $\mathcal{E}$ holds. Then, there exists a constant $C$ such that

$$\mathrm{P}\Big(\max_{1\le\alpha\le K}|R'_{\Gamma,l}(x,q_\alpha) - R^E_\Gamma(x,q_\alpha)| > \varepsilon\Big) \le 2\frac{\mathrm{vol}(N)}{S_1}\Big(\frac{2}{\delta}\Big)^n e^{-C\,l\,h^m\varepsilon^2},$$

which can be shown using Bernstein's inequality for $\frac{1}{l}\sum_{i=1}^{l} W_i - \mathbb{E}[W_i]$ where $W_i = \frac{\Gamma(d_N(q_\alpha,Y_i))k_h(d_M(x,X_i))}{\mathbb{E}[k_h(d_M(x,X))]}$ together with a union bound over the elements in the covering $\mathcal{N}_\delta$ using

$$|W_i| \le \frac{b}{a}\frac{\Gamma(\mathrm{diam}(N))}{h^m S_1 r_1^m p_{\min}}, \quad \mathrm{Var}\,W_i \le \frac{\Gamma(\mathrm{diam}(N))^2\mathbb{E}[k_h^2(d_M(x,X))]}{(\mathbb{E}[k_h(d_M(x,X))])^2} \le \frac{b}{a}\frac{\Gamma(\mathrm{diam}(N))^2}{h^m S_1 r_1^m p_{\min}},$$

where we used Proposition 1 to lower bound $\mathrm{vol}(B(x,h\,r_1))$ for small enough $h$. Third, we get for the third term using again Lemma 2,

$$|R^E_\Gamma(x,q_\alpha) - R^E_\Gamma(x,q)| \le 2\Gamma'(\mathrm{diam}(N))d_N(q,q_\alpha) \le 2\Gamma'(\mathrm{diam}(N))\delta.$$

Last, we have to bound the approximation error $R^E_\Gamma(x,q) - R'_\Gamma(x,q)$, Under the continuity assumption on the joint density $p(x,y)$ we can use Proposition 2. For every $x \in M\backslash\partial M$ we get,

$$\lim_{h\to 0}\int_M k_h(d_M(x,z))p(z,y)dV(z) = C_x\,p(x,y), \quad \lim_{h\to 0}\int_M k_h(d_M(x,z))p(z)dV(z) = C_x p(x),$$

where $C_x > 0$. Thus with

$$f_h = \int_M k_h(d_M(x,z))p(z,y)dV(z), \quad g_h = \int_M k_h(d_M(x,z))p(z)dV(z),$$

we get for every $x \in M\backslash\partial M$,

$$\lim_{h\to 0}\Big|\frac{f_h}{g_h} - \frac{f}{g}\Big| \le \lim_{h\to 0}\frac{|f_h - f|}{g_h} + \lim_{h\to 0}f\frac{|g_h - g|}{g\,g_h} = 0,$$

where we have used $g_h \ge aS_1 r_1 p_{\min} > 0$ and $g = C_x p(x) > 0$. Moreover, using results from the proof of Proposition 2 one can show $f_h < C$ for some constant $C$. Thus $f_h/g_h < C$ for some constant and $f_h/g_h \to f/g$ as $h \to 0$. Using the dominated convergence theorem we thus get

$$\lim_{h\to 0}R^E_\Gamma(x,q) = \lim_{h\to 0}\frac{\mathbb{E}[\Gamma(d_N(q,Y))k_h(d_M(x,X))]}{\mathbb{E}[k_h(d_M(x,X))]} = \int_N \Gamma\big(d_N(q,y)\big)\frac{p(x,y)}{p(x)}dy = R'_\Gamma(x,q).$$

For the case where the joint density is Lipschitz continuous one gets using Proposition 2, $R^E_\Gamma(x,q) = R'_\Gamma(x,q) + O(h)$.

In total, there exist constants $A, B, C, D_1, D_2$, such that for sufficiently small $h$ one has with probability $1 - Ae^{B\,n\log(\frac{1}{\delta})-Clh^m\varepsilon^2}$,

$$\sup_{q\in N}|R'_{\Gamma,l}(x,q) - R^E_\Gamma(x,q)| \le 2D_1\delta + \varepsilon.$$

With $\delta = l^{-s}$ for some $s > 0$ one gets convergence if $\frac{lh^m}{\log l} \to \infty$ together with $\lim_{h\to 0}R^E_\Gamma(x,q) = R'_\Gamma(x,q)$. For the case where $p(\cdot,y)$ is Lipschitz continuous for all $y \in N$ we have $R^E_\Gamma(x,q) = R'_\Gamma(x,q) + O(h)$ and can choose $s$ large enough so that the bound from the approximation error dominates the one of the covering. Under the condition $\frac{lh^m}{\log l} \to \infty$ the probabilistic bound is summable in $l$ which yields almost sure convergence by the Borel-Cantelli-Lemma. The optimal rate in the Lipschitz continuous case is then determined by fixing $h$ such that both terms of the bound are of the same order. $\square$

## Acknowledgments

We thank Florian Steinke for helpful discussions about relations between generalized kernel estimators and structured output learning. This work has been partially supported by the Cluster of Excellence MMCI at Saarland University.

## Footnotes

[1] A metric space is separable if it contains a countable dense subset.

[2]In some cases the set of all local minimizers is denoted as the Frechét mean set and the Frechét mean is called unique if there exists only one global minimizer.

[3] The kernel $k_N$ induces a (semi-)metric $d_N$ on $N$ via: $d_N^2(p, q) = k_N(p, p) + k_N(q, q) - 2k_N(p, q)$.

[4]The set of points where there the minimizing geodesic is not unique, the so called cut locus, has measure zero and therefore plays no role in the optimization.

# References

[1] I. Tsochantaridis, T. Joachims, T. Hofmann, and Y. Altun. Large margin methods for structured and interdependent output variables. *JMLR*, 6:1453–1484, 2005.

[2] J. Weston, G. BakIr, O. Bousquet, B. Schölkopf, T. Mann, and W. S. Noble. Joint kernel maps. In *Predicting Structured Data*, pages 67–84. MIT Press, 2007.

[3] E. Ricci, T. De Bie, and N. Cristianini. Magic moments for structured output prediction. *JMLR*, 9:2803–2846, 2008.

[4] K.V. Mardia and P.E. Jupp. *Directional statistics*. Wiley New York, 2000.

[5] Inam Ur Rahman, Iddo Drori, Victoria C. Stodden, David L. Donoho, and Peter Schroder. Multiscale representations for manifold-valued data. *Multiscale Modeling and Simulation*, 4(4):1201–1232, 2005.

[6] B. C. Davis, P. T. Fletcher, E. Bullitt, and S. Joshi. Population shape regression from random design data. *Computer Vision, 2007. ICCV 2007. IEEE 11th International Conference on*, pages 1–7, 2007.

[7] F. Steinke and M. Hein. Non-parametric regression between Riemannian manifolds. In *Advances in Neural Information Processing Systems (NIPS) 21*, pages 1561 – 1568, 2009.

[8] P. T. Fletcher, S. Venkatasubramanian, and S. Joshi. The geometric median on Riemannian manifolds with application to robust atlas estimation. *NeuroImage*, 45:143 – 152, 2009.

[9] C. G. Small. A survey of multidimensional medians. *International Statistical Review*, 58:263–277, 1990.

[10] D. Blackwell and M. Maitra. Factorization of probability measures and absolutely measurable sets. *Proc. Amer. Math. Soc.*, 92(2):251–254, 1984.

[11] R. Bhattacharya and V. Patrangenaru. Large sample theory of intrinsic and extrinsic sample means on manifolds I. *Ann. Stat.*, 31(1):1–29, 2003.

[12] H. Karcher. Riemannian center of mass and mollifier smoothing. *Communications on Pure and Applied Mathematics*, 30:509–541, 1977.

[13] W. Kendall. Probability, convexity, and harmonic maps with small image. I. Uniqueness and fine existence. *Proc. London Math. Soc.*, 61(2):371–406, 1990.

[14] P. Indyk. Sublinear time algorithms for metric space problems. In *Proceedings of the 31st Symposium on Theory of computing (STOC)*, pages 428 – 434, 1999.

[15] L. Györfi, M. Kohler, A. Krzyżak, and H. Walk. *A Distribution-Free Theory of Nonparametric Regression*. Springer, New York, 2004.

[16] W. Greblicki and M. Pawlak. *Nonparametric System Identification*. Cambridge University Press, Cambrige, 2008.

[17] B. Pelletier. Nonparametric regression estimation on closed Riemannian manifolds. *J. of Nonparametric Stat.*, 18:57–67, 2006.

[18] S. Dabo-Niang and N. Rhomari. Estimation non parametrique de la regression avec variable explicative dans un espace metrique. *C. R. Math. Acad. Sci. Paris*, 1:75–80, 2003.

[19] D. P. Bertsekas. *Nonlinear Programming*. Athena Scientific, Belmont, Mass., 1999.

[20] M. Hein. Uniform convergence of adaptive graph-based regularization. In G. Lugosi and H. Simon, editors, *Proc. of the 19th Conf. on Learning Theory (COLT)*, pages 50–64, Berlin, 2006. Springer.

[21] N. Glick. Consistency conditions for probability estimators and integrals of density estimators. *Utilitas Math.*, 6:61–74, 1974.

